# Optoelectronic Implementation of a FitzHugh-Nagumo Neural Model

**Alexandre R.S. Romariz** *, **Kelvin Wagner**
Optoelectronic Computing Systems Center
University of Colorado, Boulder, CO, USA 80309-0425
`romariz@colorado.edu`

## Abstract

An optoelectronic implementation of a spiking neuron model based on the FitzHugh-Nagumo equations is presented. A tunable semiconductor laser source and a spectral filter provide a nonlinear mapping from driver voltage to detected signal. Linear electronic feedback completes the implementation, which allows either electronic or optical input signals. Experimental results for a single system and numeric results of model interaction confirm that important features of spiking neural models can be implemented through this approach.

## 1 Introduction

Biologically-inspired computation paradigms take different levels of abstraction when modeling neural dynamics. The production of action potentials or spikes has been abstracted away in many rate-based neurodynamic models, but recently this feature has gained renewed interest [1, 2]. A computational paradigm that takes into account the timing of spikes (instead of spike rates only) might be more efficient for signal representation and processing, especially at short time windows [3, 4, 5].

Optics technology provides high bandwidth and massive parallelism for information processing. However, the implementation of digital primitives have not as yet proved competitive against the scalability and low power operation of digital electronic gates. It is then natural to explore the features of optics for different computational paradigms. Artificial neural networks promise an excellent match to the capabilities of optics, as they emphasize simple analog operations, parallelism and adaptive interconnection[6, 7, 8, 9].

Optical implementations of Artificial Neural Networks have to deal with the problem of representing the nonlinear activation functions that define the input-output mappings for each neuron. Although nonlinear optics has been suggested for implementing neurons, hybrid optoelectronic systems, where the task of producing nonlinearity is given to the electronic circuits, may be more practical [10, 11]. In the case of pulsing neurons, the task seems more difficult still, for instead of a nonlinear static map we are required to implement a nonlinear dynamical system. Several possibilities for the implementation of pulsed optical neurons can be considered, including smart pixel pulsed electronic circuits with op-

tical inputs [12], pulsing laser cavity feedback dynamics [13] and competitive-cooperative phosphor feedback [14].

In this paper we demonstrate and evaluate an optoelectronic implementation of an artificial spiking neuron, based on the FitzHugh-Nagumo equations. The proposed implementation uses wavelength tunability of a laser source and a birefringent crystal to produce a nonlinear mapping from driving voltage to detected optical output [15]. Linear electronic feedback to the laser drive current completes the physical implementation of this model neuron. Inputs can be presented optically or electronically, and output signals are also readily available as optical or electronic pulses.

This work is organized as follows. Section 2 reviews the FitzHugh-Nagumo equations and describes the particular optoelectronic spiking neuron implementation we propose here. In Section 3 we analyze and illustrate dynamical properties of the model. Experimental results of the optoelectronic system implementing one model are presented in Section 4. Numeric results that illustrate features of the interaction between models are shown in Section 5.

## 2 Modified FN Neural Model and optoelectronic implementation

The FitzHugh-Nagumo neuron model [16, 17] is appealing for physical implementation, as it is fairly simple and completely described by a pair of coupled differential equations:

$$
\begin{aligned}
\tau_v \dot{v}(t) &= f[v(t)] - w(t) + u(t) \\
\tau_w \dot{w}(t) &= Av(t) - w(t) + B
\end{aligned}
\tag{1}
$$

where $v$ is an excitable state variable that exhibits bi-stability as a result of the nonlinear $f[v]$ term, and $w$ is a linear recovery variable, bringing the neuron back to a resting state. In the original model proposal, $f[v]$ is a third-degree polynomial[16, 17]. This model has been previously implemented in CMOS integrated electronics [18].

In optical implementation of neural networks, the required nonlinear functions are usually performed through electronic devices, with adaptive linear interconnection done in the optical domain. We here explore the possibility of optical implementation of the required nonlinear function $f[x]$ by using the nonlinear response of linear optical systems to variations of the wavelength.

Consider a birefringent material placed between crossed polarizers. Even though propagation of the field through the material is a linear phenomenon (a linear phase difference among orthogonal polarization components is generated), the output power as a function of incident wavelength is sinusoidal, according to

$$
v_{det} = RI_{\text{det}} = R\mathcal{R}P(i)\sin^2\left(\frac{\pi D}{\lambda(i)}\right)
\tag{2}
$$

where $R$ is the transimpedance gain of the detector amplifier, $\mathcal{R}$ is the responsivity (in A/W), $P(i)$ is the optical power incident on the detector, which is a function of the laser drive current $i$, $D$ is the optical path difference (OPD) resulting from propagation through the birefringent material and $\lambda(i)$ is the laser wavelength.

In semiconductor lasers, and Vertical Cavity Surface Emitting Lasers (VCSELs) in particular, an input current $i$ produces a small modulation in the radiation wavelength $\lambda(i)$. Linearizing the $1/\lambda$ variation in Equation 2, we find a nonlinear mapping from driving voltage to detected signal:

$$
v_{det} = f(v) = G(v)\sin^2\left(\frac{\pi(v - V_\phi)}{V_T}\right)
\tag{3}
$$

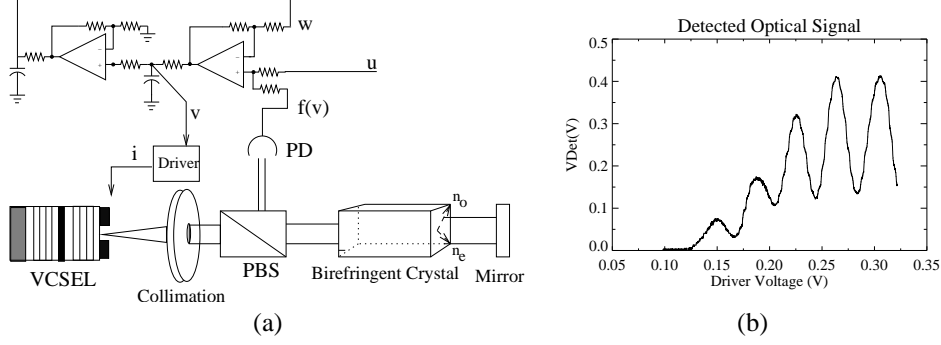

Figure 1: **a** Experimental setup for the wavelength-based nonlinear oscillator, with simplified view of the electronic feedback. **b** Experimental evidence of nonlinear mapping from driver voltage to detected signal (open loop), as a result of wavelength modulation as well as laser threshold and saturation.

where $v$ is the driving voltage (linearly converted to an input current $i$ through the driver transconductance) and the function $G(v)$ includes all conversion factors in the detection process, as well as nonlinear phenomena such as laser threshold and saturation.

A simple nonlinear feedback loop can now be established, by feeding the detected signal back to the driver. This basic arrangement has been used to investigate chaotic behavior in delayed-feedback tunable lasers [15] . It is used here as the nonlinearity for an optical self-pulsing mechanism in order to implement neural-like pulses based on the following dynamical system

$$\begin{aligned} \tau_v \dot{v}(t) &= f[v(t)] - v(t) - w(t) + u(t) \\ \tau_w \dot{w}(t) &= Av(t) - w(t) + B \end{aligned} \tag{4}$$

Again $v$ is a fast state variable, and $w$ a relatively slow recovery variable, so that $\tau_v \ll \tau_w$. The experimental setup is shown in Figure 1a. Light from the tunable source is collimated and propagates through a piece of birefringent crystal. The crystal fast and slow axis are at 45 degrees to the polarizer and analyzer passing axis. The effective propagation length through the crystal (and corresponding wavelength selectivity) is doubled with the use of a mirror. A polarizing beam splitter acts as both polarizer and analyzer. A simplified view of the electronic feedback is also shown. Leaky integrators and linear analog summations implement the linear part of Equation 4, while the nonlinear response (in intensity) of the optical filter implements $f(v)$.

A VCSEL was used as tunable laser source. These vertical-cavity semiconductor lasers have, when compared to edge-emitting diode lasers, larger separation between longitudinal modes, more circularly-symmetric beams and lower fabrication costs [19]. As the input current is increased, the heating of the cavity red-shifts the resonant wavelength [20], and this is the main mechanism we are exploring for wavelength modulation.

An experimental verification of the expected sinusoidal variation of detected power with modulation voltage is given in Figure 1b. A slow (800Hz) modulation ramp was applied to the driver, and the detected power variation was acquired. From this information, the static transfer function shown in the right part of the figure was calculated. Unlike the experiment with a DBR laser diode reported by Goedgebuer *et al.* [15], it is apparent that current modulation is affecting not only wavelength (and hence effective optical path difference among polarization components) but overall output power as well. Modulation depth is limited (non-zero troughs in the sinusoidal variation), which we attribute to the multiple transverse modes that the device supports. However, as we are going to be operating near

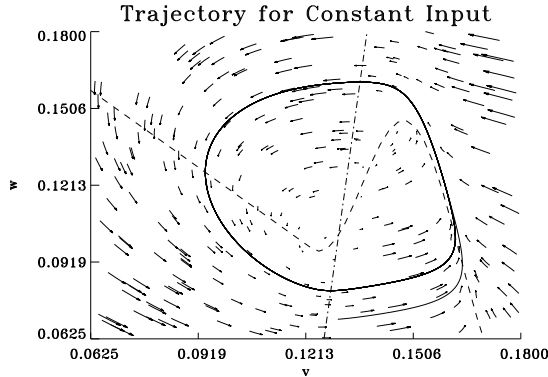

Figure 2: Continuous line: trajectory of the system under strong input, obtained by numeric integration ($4^{th}$-order Runge-Kutta) of Equation 4. Arrows represent the strength of the derivatives at a particular point in state space. Dashed line: nullcline $\dot{v} = 0$. Dash-dotted line: nullcline $\dot{w} = 0$. Stability analysis show that the equilibrium point where the nullclines meet is unstable, so the limit cycle is the sole attractor. Parameters $\tau_w = 10\tau_v$, $A = 10$, $B = -1.2$, $u = 0.22$V,$V_T = 0.039$V,$V_\phi = 0.123$V.

the first maximum (see Section 3), the power variation over successive maxima should not affect the dynamical properties of the closed-loop system. The relatively smooth curve obtained indicates that no mode hops occurred for this driving current range, which was indeed confirmed with Optical Spectrum Analyzer measurements.

## 3    Simulations

FitzHugh-Nagumo models are known to have so-called class II neural excitability (see [21] for a review). This class is characterized by an Andronov-Hopf bifurcation for increasing excitation, and exhibits some dynamical phenomena that are not present in integrate-and-fire dynamics. For equal intensity input pulses, integrators will respond maximally to the pulse train with lowest inter-spike interval. Class II neurons have resonant response to a range of input frequencies. There are non-trivial forms of excitation in resonator models that are not matched by integrators: the former can produce a spike at the end of an inhibitory pulse, and conversely, can have a limit cycle condition interrupted (with the system recovering to rest) by an excitatory pulse.

We have verified that these characteristics are maintained in the modified optical model, despite the use of a sinusoidal nonlinearity instead of the original $3^{rd}$ degree polynomial function. Stability analysis based on the Jacobian of the dynamical system (Equation 4) shows an Andronov-Hopf bifurcation, as in the original model. Limit cycle interruption through exciting pulses is shown in Section 5.

Figure 2 shows a typical limit-cycle trajectory, for parameter values that match conditions of the experiment reported in Section 4. Parameters were chosen so that a typical excursion in modulation voltage goes from the dead zone (below the lasing threshold) to around the first peak in the nonlinear detector transfer function. This is an interesting choice because the optical output is only present during spiking, and can be used directly as an input to other optoelectronic neurons.

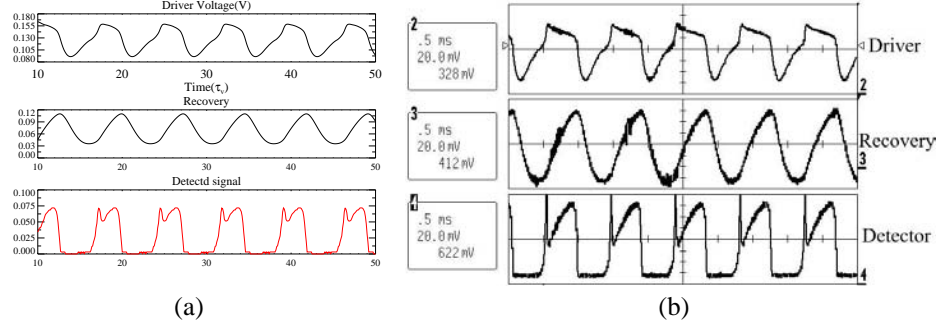

Figure 3: Dynamical system response to strong constant input. **a** Simulation results. Parameters as in Figure 2. **b** Experimental results. Parameters: $\tau_v = 0.1$ms, $\tau_w = 1$ms.

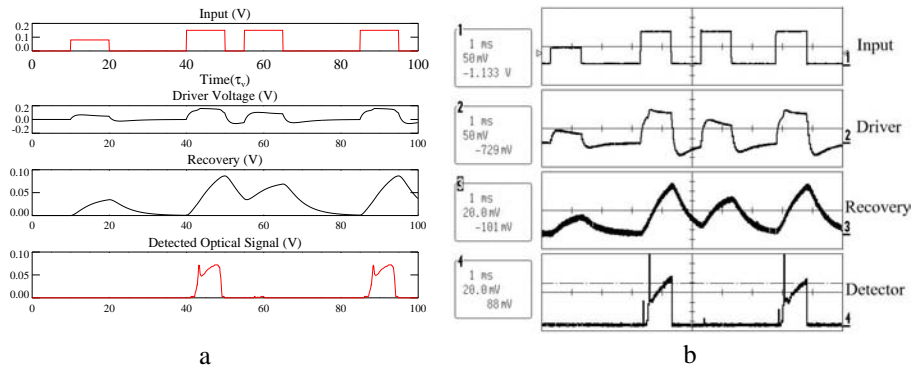

Figure 4: **(a)**: Simulated response to a train of pulses. Parameters as in Figure 2. **(b)**: Experimental Results. Parameters as in Figure 3.

## 4   Experimental Results

Figure 3 presents a comparison between simulated waveforms for the various dynamic variables involved (as the system performs the trajectory depicted in Figure 2) and the experimental results obtained with the system described in Figure 1, revealing a good agreement between simulated and experimental waveforms. The double-peak in the optical variable can be understood by following the trajectory indicated in Figure 2, bearing in mind the non-monotonic mapping from driver voltage to detected signal. The decrease in driver voltage observed as the recovery variable $w$ increases produces initially an increase in detected power, and thus the second, broader peak at the end of the cycle.

The production of sustained oscillations for constant input is one of the desired characteristics of the model, but in a network, neurons will mostly communicate through their pulsed output. The response of the system to pulsed inputs can be seen in Figure 4. The output optical signal response is all-or-none, but sub-threshold integration of weak inputs is being performed, as the waveform for driver voltage shows in the first pulse. As $w$ slowly returns to 0, a new excitation just after a pulse is less likely, which can be seen at the response to the third pulse. The experimentally observed waveforms agree with the simulations, though details of the pulsing in the optical output are different.

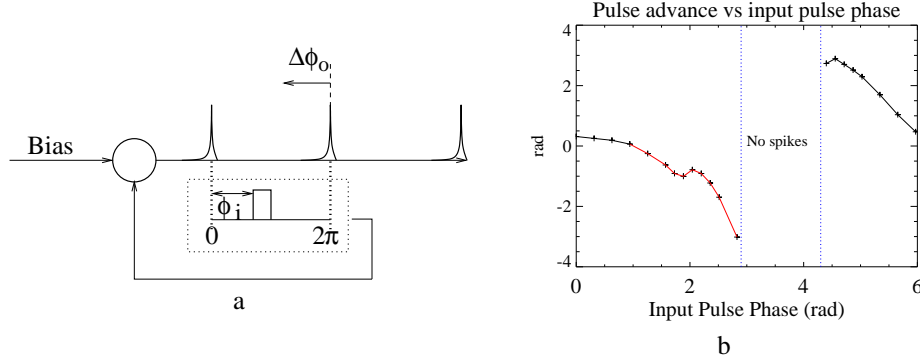

Figure 5: Numeric illustration of the effect of input timing on the advance of the next spike, in the modified FitzHugh-Nagumo system. **a**: Schematic view of simulation. See text for details. **b**: Phase advance as a function of input phase. Bias 0.103V. Input pulse height 10 mV, duration $10\tau_v$. Dynamic system parameters as in Figure 2.

## 5 Coupling

One of the main motivations for using optical technology in neural network implementation is the possibility of massive interconnection, and so the definition of coupling techniques, and the study of adaptation algorithms compatible with the dynamical properties of the experimentally-demonstrated oscillators are the current focus of this research.

The most elegant optical implementation of adaptive interconnection is through dynamic volume holography[6, 11], but that requires a set of coherent optical signals, not what we have with an array of pulse emitters. In contrast, the matrix-vector multiplier architecture allows parallel interconnection of incoherent optical signals, and has been used to demonstrate implementations of the Hopfield model [7] and Boltzman machines [9].

An interesting aspect of the coupled dynamics in oscillators exhibiting class II excitability is that the timing of an input pulse can result in advance or retardation of the next spike [22]. This is potentially relevant for hardware implementation, as the excitatory (i.e., inducing an early spike) or inhibitory character of the connection might be controlled without changing signs of the coupling strength.

In Figure 5 we show a simulation illustrating the effect of input pulse timing in advancing the output spike. A constant input to a model neuron (Equation 4) was maintained, producing periodic spiking. A second, positive, pulsed input was activated in between spikes, and the effect of this coupling on the advance or retardation of the next spike was verified as the timing of the input was varied. A region of output spike retardation ($\Delta\phi_o < 0$) with excitatory pulsed input can be seen. Even more interesting, for phases around $\pi$ rad relative to the latest spike, the excitatory pulse can terminate periodic spiking altogether.

This phenomenon is seen in detail in Figure 6, where both the time waveforms and statespace trajectories are shown. For this particular condition, the equilibrium point of the system is stable. When correctly timed, the short excitatory pulse forces the system out of its limit cycle, into the basin of attraction of the stable equilibrium, hence stopping the periodic spiking. As the individual models used in this simulations were shown to match experimental implementations in Section 4, we expect to observe the same kind of effect in the coupling of the optoelectronic oscillators.

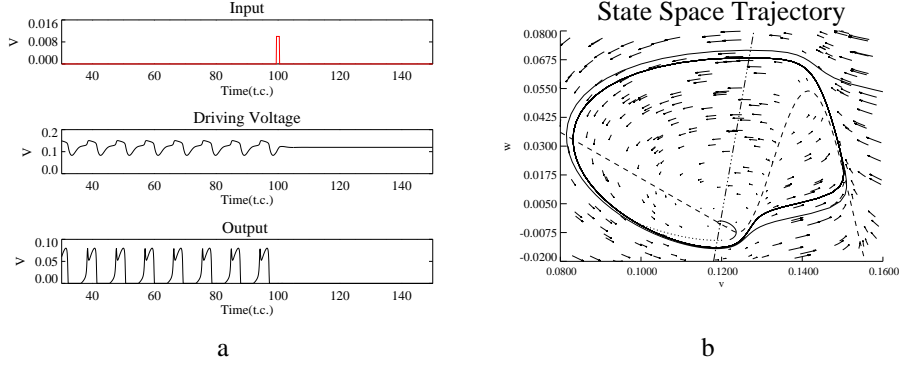

Figure 6: **(a)**: Simulated response illustrating return to stability with excitatory pulse. $u = 0.116$. Other parameters as in Figure 2. **(b)**: Same results in state space. Continuous line: Unperturbed trajectory. Dotted Line: Trajectory during excitatory pulse.

# 6 Ongoing work and conclusions

Implementation of a modified FN neuron model with a nonlinear transfer function realized with a wavelength-tuned VCSEL source, a linear optical spectral filter and linear electronic feedback was demonstrated. The system dynamical behavior agrees with simulated responses, and exhibits some of the basic features of neuron dynamics that are currently being investigated in the area of spiking neural networks.

Further experiments are being done to demonstrate coupling effects like the ones described in Section 5. In particular, the use of external optical signals directly onto the detector to implement optical coupling has been demonstrated. Feedback circuit simplification is another important aspect, since we are interested in implementing large arrays of spiking neurons. With enough detection gain, Equation 4 should be implementable with simple RLC circuits, as in the original work by Nagumo[17].

Results reported here were obtained at low frequency (1-100 KHz), limited by amplifier and detector bandwidths. With faster electronics and detectors, the limiting factor in this arrangement would be the time constant for thermal expansion of the VCSEL cavity, which is around $1\mu s$. Pulsing operation at 1.2 MHz has been obtained in our latest experiments.

Even faster operation is possible when using the internal dynamics of wavelength modulation itself, instead of external electronic feedback. In addition to the thermally-induced modulation of wavelength, carrier injection modifies the index of refraction of the active region directly, which results in an opposite wavelength shift. By using this carrier injection effect to implement the recovery variable, feedback electronics is simplified and a much faster time constant controls the model dynamics. Optical coupling of VCSELs has the potential to generate over 40GHz pulsations [23]. Our goal is to investigate those optical oscillators as a technology for implementing fast networks of spiking artificial neurons.

**Acknowledgments**

This research is supported in part by a Doctorate Scholarship to the first author from the Brazilian Council for Scientific and Technological Development, CNPq.

## Footnotes

*On leave from the Electrical Engineering Department, University of Brasília, Brazil

# References

[1] F. Rieke, D. Warland, R.R. von Steveninck, and W. Bialek. *Spikes: Exploring the Neural Code*. MIT Press, Cambridge, USA, 1997.

[2] T.J. Sejnowski. Neural pulse coding. In W. Maass and C.M. Bishop, editors, *Pulsed Neural Networks*, Cambridge, USA, 1999. The MIT Press.

[3] W. Maass. Lower bounds for the computational power of spiking neurons. *Neural Computation*, 8:1–40, 1996.

[4] J.J. Hopfield. Pattern recognition computation using action potential timing for stimulus representation. *Nature*, 376:33–36, 1995.

[5] R. van Rullen and S.J. Thorpe. Rate coding versus temporal order coding: what the retinal ganglion cells tells the visual cortex. *Neural Computation*, 13:1255–1283, 2001.

[6] D. Psaltis, D. Brady, and K. Wagner. Adaptive optical networks using photorefractive crystals. *Applied Optics*, 27(9):334–341, May 1988.

[7] N.H. Farhat, D. Psaltis, A. Prata, and E. Paek. Optical implementation of the Hopfield model. *Applied Optics*, 24:1469–1475, 1985.

[8] S. Gao, J. Yang, Z. Feng, and Y. Zhang. Implementation of a large-scale optical neural network by use of a coaxial lenslet array for interconnection. *Applied Optics*, 36(20):4779–4783, 1997.

[9] A.J. Ticknor and H.H. Barrett. Optical implementation of Boltzmann machines. *Optical Engineering*, 26(1):16–21, January 1987.

[10] K.S. Hung, K.M. Curtis, and J.W. Orton. Optoelectronic implementation of a multifunction cellular neural network. *IEEE Transactions on Circuits and Systems II*, 43(8):601–608, August 1996.

[11] K. Wagner and T.M. Slagle. Optical competitive learning with VLSI liquid-crystal winner-take-all modulators. *Applied Optics*, 32(8):1408–1435, March 1993.

[12] K. Hynna and K. Boahen. Space-rate coding in an adaptive silicon neuron. *Neural Networks*, 14(6):645–656, July 2001.

[13] F. Di Theodoro, E. Cerboneschi, D. Hennequin, and E. Arimondo. Self-pulsing and chaos in an extended-cavity diode laser with intracavity atomic absorber. *International Journal of Bifurcation and Chaos*, 8(9), September 1998.

[14] J.L. Johnson. All-optical pulse generators for optical computing. In *Proceedings of the 2002 International Topical Meeting on Optics in Computing*, pages 195–197, Taipei, Taiwan, 2002.

[15] J. Goedgebuer, L. Larger, and H.Porte. Chaos in wavelength with a feedback tunable laser diode. *Physical Review E*, 57(3):2795–2798, March 1998.

[16] R.FitzHugh. Impulses and physiological states in models of nerve membrane. *Biophysical Journal*, 1:445–466, 1961.

[17] J. Nagumo, S. Arimoto, and S. Yoshizawa. An active pulse transmission line simulating nerve axon. *Proceedings of the IRE*, 50:2061–2070, 1962.

[18] B. Linares-Barranco, E. Sánchez-Sinencio, A. Rodríguez-Vázquez, and J.L. Huertas. A CMOS implementation of FitzHugh-Nagumo neuron model. *IEEE Journal of Solid-State Circuits*, 26(7):956–965, July 1991.

[19] A. Yariv. *Optical Electronics in Modern Communications*. Oxford University Press, New York, USA, fifth edition, 1997.

[20] W. Nakwaski. Thermal aspects of efficient operation of vertical-cavity surface-emitting lasers. *Optical and Quantum Electronics*, 28:335–352, 1996.

[21] E.M. Izhikevich. Neural excitability, spiking and bursting. *International Journal of Bifurcation and Chaos*, 2000.

[22] E.M. Izhikevich. Weakly pulse-coupled oscillators, FM interactions, synchronization, and oscillatory associative memory. *IEEE Transactions on Neural Networks*, 10(3):508–526, May 1999.

[23] C.Z. Ning. Self-sustained ultrafast pulsation in coupled vertical-cavity surface-emitting lasers. *Optics Letters*, 27(11):912–914, June 2002.
